# Keeping flexible active contours on track using Metropolis updates

**Trausti T. Kristjansson**
University of Waterloo
ttkristj@uwaterloo.ca

**Brendan J. Frey**
University of Waterloo
frey@uwaterloo.ca

## Abstract

Condensation, a form of likelihood-weighted particle filtering, has been successfully used to infer the shapes of highly constrained "active" contours in video sequences. However, when the contours are highly flexible (*e.g.* for tracking fingers of a hand), a computationally burdensome number of particles is needed to successfully approximate the contour distribution. We show how the Metropolis algorithm can be used to update a particle set representing a distribution over contours at each frame in a video sequence. We compare this method to condensation using a video sequence that requires highly flexible contours, and show that the new algorithm performs dramatically better that the condensation algorithm. We discuss the incorporation of this method into the "active contour" framework where a shape-subspace is used constrain shape variation.

## 1  Introduction

Tracking objects with flexible shapes in video sequences is currently an important topic in the vision community. Methods include curve fitting [9], layered models [1, 2, 3], Bayesian reconstruction of 3-D models from video[6], and active contour models [10, 14, 15].

Fitting curves to the outlines of objects has been attempted using various methods, including "Snakes" [8, 9], where an energy function is minimized so as to find the best fit. As with other optimization methods, this approach suffers from local maxima. This problem is amplified when using real data where edge noise can prevent the fit of the contour to the desired object outline.

In contrast, Blake *et al.* [10] introduced a probabilistic framework for curve fitting and tracking. Instead of proposing one single best fit for the contour, a probability distribution over contours is found. The distribution is represented as a particle set where each particle represents one contour shape. Inference in these "active contour" models is accomplished using particle filtering.

In the "active contour" method, a probabilistic dynamic system is used to model the distribution over the outline of the object (the contour) $Y_t$ and the observations $Z_t$ at time $t$. Tracking is performed by inference in this model.

The outline of an object is tracked through successive frames in a video by using a particle

(a)                                    (b)

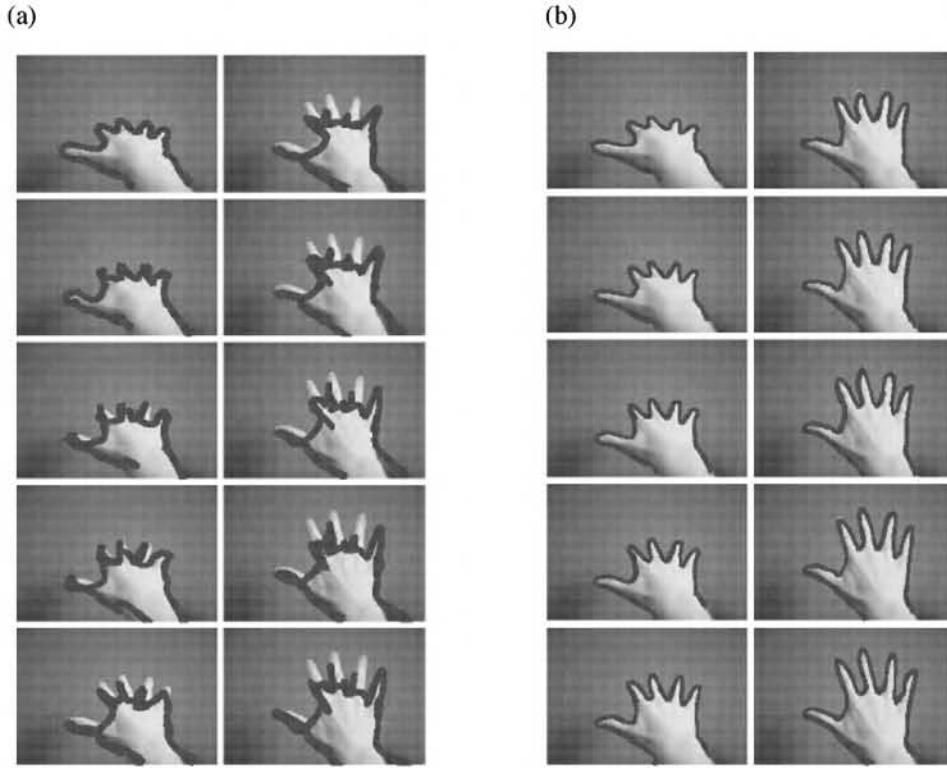

Figure 1: (a) Condensation with Gaussian dynamics (result for best $\sigma = 2$ shown) applied to a video sequence. The 200 contours corresponding to 200 particles fail to track the complex outline of the hand. The pictures show every 24th frame of a 211-frame sequence. (b) Metropolis updates with *only 12 particles* keep the contours on track. At each step, 4 iterations of Metropolis updates are applied with $\sigma = 3$.

distribution. Each particle $x_n$ represents single contour $Y$ [1] that approximates the outline of the object. For any given frame, a set of particles represents the probability distribution over positions and shapes of an object.

In order to find the likelihood of an observation $Z_t$, given a particle $x_n$, lines perpendicular to the contour are examined and edges are detected. A variety of distributions can be used to model the likelihood of the edge positions along each line. We assume that the position of the edge belonging to the object is drawn from a Gaussian with mean position at the intersection of the contour and the measurement line $Y(s_m)$ and the positions of the other edges are drawn from a Poisson distribution. The observation likelihood for a single measurement line $z_m$ can be simplified to [10]

$$p(z_m|x_n) \propto 1 + \frac{1}{\sqrt{2\pi}\sigma_{ml}\alpha} \sum_j \exp\left[-\frac{|z_{m,j} - B(s_m)x_n|^2}{2\sigma_{ml}^2}\right] \qquad (1)$$

where $z_{m,j}$ denotes the coordinates of an edge on measurement line $m$, and $B(s_m)x_n = Y_n(s_m)$ is the intersection of the contour and the measurement line (see later). $\alpha = q\lambda$

where $q$ is the probability of not observing the edge, and $\lambda$ is the rate of the Poisson process. $\sigma_{ml}$ defines the standard deviation in pixels. A multitude of measurement lines is used along the contour, and (assuming independence) the contour likelihood is

$$p(Z|x_n) = \prod_M p(z_m|x_n) \tag{2}$$

where $m \in M$ is the set of measurement lines.

As mentioned, in the condensation algorithm, a particle set is used to represent the distribution of contours. Starting from an initial distribution, a new distribution for a successive frame is produced by propagating each particle using the system dynamics $P(x_t|x_{t-1})$. Now the observation likelihood $P(Z_t|x_t)$ is calculated for each particle, and the particle set is resampled with replacement, using the likelihoods as weights. The resulting set of particles approximates the posterior distribution at time $t$ and is then propagated to the next frame.

Figure 1(a) shows the results of using condensation with 200 particles. As can be seen, the result is poor. Intuitively, the reason condensation fails is that it is highly unlikely to draw a particle that has raised control points over the four fingers, while keeping the remainder fixed. Figure 1(b) shows the result of using Metropolis updates and 12 particles (equivalent amount of computation).

## 2   Keeping contours on track using Metropolis updates

To reduce the dimensionality of the inference, a subspace is often used. For example, a fixed shape is only allowed horizontal and vertical translation. Using a subspace reduces the size of the required particle set, allowing for successful tracking using standard condensation. If the object can deform, a subspace that captures the allowed deformations may be used [15]. This increases the flexibility of the contour, but at the cost of enlarged dimensionality. In order to learn such a subspace, a large amount of training samples are used, which are supplied by hand fitting contour shapes to a large number of frames. However, even moderately detailed contours (say, the outline of a hand) will have many control points that interact in complex ways, making subspace modeling difficult or impractical.

### 2.1   Metropolis sampling

Metropolis sampling is a popular Markov Chain Monte Carlo method for problems of large dimensionality[16, 17]. A new particle is drawn from a proposal density $Q(x'; x_t)$, where in our case, $x_t$ is a particle (i.e. a set of control points) at time $t$, and $x'$ is a tentative new particle produced by perturbing a subset of the control points.

$$Q_i(x'|x_t) = \frac{1}{\sqrt{2\pi\sigma^2}} \exp\left[-\frac{(x'-x_t)^2}{2\sigma^2}\right]. \tag{3}$$

We then calculate

$$a = \frac{p(x_t'|x_{t-1})p(z_t|x_t')}{p(x_t|x_{t-1})p(z_t|x_t)} \frac{Q_i(x';x_t)}{Q_i(x_t;x')} \tag{4}$$

where $p(x_t|x_{t-1})p(z_t|x_t)$ is proportional to the posterior probability of observing the contour in that position. If $a \geq 1$ the proposed particle is accepted. If $a < 1$, it is accepted with probability $a$. Since $Q$ is symmetric, the second factor $Q(x';x_t)/Q(x_t;x') = 1$.

Metropolis sampling can be used in the framework of particle propagation in two ways. It can either be used to fit splines around contours of a training set that is used to construct a shape subspace, e.g. by PCA, or it can also be used to refine the shapes of the subspace to the actual data during tracking.

## 2.2 B-splines

B-splines or basis function splines are parametric curves, defined as follows:

$$Y(s) = B(s)C \tag{5}$$

where $Y(s)$ is a two dimensional vector consisting of the 2-D coordinates of a point on the curve, $B(s)$ is a matrix of polynomial basis functions, and $C$ is a vector of control points. In other words, a point along the curve $Y(s)$ is a weighted sum of the values of the basis functions $B(s)$ for a particular value of $s$, where the weights are given by the values of $C$. The basis functions of b-splines have the characteristic that they are non-zero over a limited range of $s$. Thus a particular control point will only affect a portion of the curve. For *regular* b-splines of order 4 (the basis functions are 3rd degree polynomials), a single control point will only affect $Y(s)$ over a range of $s$ of length 4. Conversely, for particular $s_m$ ($m : s_m \in SupportOf(x_i)$, where $i$ indexes the component of x that has been altered), $Y(s_m)$ is affected by at most 4 control points (fewer towards the ends).

As mentioned before, a detailed contour can have a large number of control points, and thus high dimensionality and so it is common to use a subspace. In this case $C$ can be written as $C = Wx + C_0$ where $W$ defines a linear subspace and $C_0$ is the template of control points, and $x$ represents perturbations from the template in the subspace.

In this work we examine unconstrained models, where no prior knowledge about the deformations or dynamics of the object are presumed. In this case $W$ is the identity matrix, $C_0 = 0$, and $x$ are the actual coordinates of the control points. This allows the contour to deform in any way.

## 2.3 Metropolis updates in condensation

The new algorithm consists of two steps: a Metropolis step, followed by a resampling step.

1. Iterate over control points:
   - For one control point at a time, draw a proposal particle by drawing a new control point $x_i'$ from a 2-D Gaussian centered at the current control point $x_{t,i}$, Eq. (3), and keeping all others unchanged.
   - Calculate the observation likelihood for the new control point, Eq. (2).
   - Calculate $a$ (Eq. 4) and reject or accept the new particle

2. Resample

3. Get next image in video

If the particle distribution at $t - 1$ reflects $P(x_{t-1}|Z_1, \ldots, Z_{t-1})$, the Metropolis updates will converge to $P(x_t|Z_1, \ldots, Z_t)$ [16].

As mentioned above, the affect of altering the position of a control point is to change the shape of the contour locally since the basis functions have limited support. Thus, when evaluating $p(x_t'|x_{t-1})p(z_t|x_t')$ for a proposed particle, we only need to reexamine measurement lines and evaluate $p(z_{m,t}|x_{n,t}')$ for lines in the effected interval and similarly for $p(x_{n,t}'|x_{n,t-1})$. This allows for an efficient algorithm implementation.

The computation $c_M$ required to update a single particle using metropolis, compared to condensation is $c_M = o \cdot it \cdot c_C$ where $o$ is the order of the b-spline, $it$ is the number of iterations, and $c_C$ is the number of computations required to update a particle using condensation. Thus, in the case of fourth order splines such as the ones we use, the increase in computation for a single particle is only four for a single iteration, and eight for two iterations. However, we have seen that far fewer particles are required.

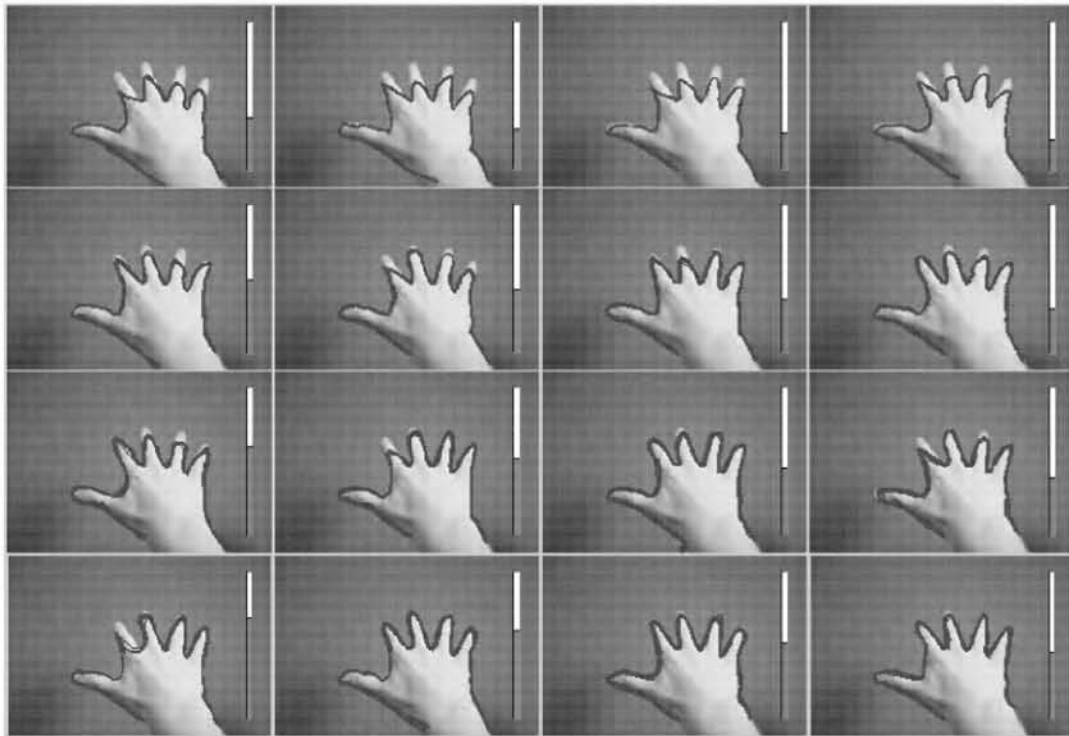

Figure 2: The behavior of the algorithm with Metropolis updates is shown at frame 100 ($t = 100$) as a function of iterations and $\sigma$. The columns, show, from left to right, 1,2,4 and 8 iterations, and the rows, from top to bottom show $\sigma = \{1, 2, 3, 4\}$. The rejection ratio (i.e. the ratio of rejected proposal particles to the total number of proposed particles) is shown as a bar on the right side of each image.

## 3  Results

We tested our algorithm on the video sequence shown in Figure 1. The contour had 56 2-D control points i.e a state space of 112 dimensions. Such high dimensionality is required for the detailed contours required to properly outline the fingers of the hand.

The results presented are for relatively noise free data, i.e. free from background clutter. This allows us to contrast the performance of using Metropolis updates and standard condensation, for the scenarios of interest, i.e. the learning of subspace models and contour refinement.

Figure 1(b) shows the results for the Metropolis updates for 12 particles, 4 iterations and $\sigma = 3$. The figure shows every 24th frame from frame 1 to frame 211. The outline of the splayed fingers is tracked very successfully.

Figure 1(a) shows every 24th frame for the condensation algorithm of equivalent complexity, using 200 particles and $\sigma = 2$. This value of $\sigma$ gave the best results for 200 particles. As can be seen, the little finger is tracked moderately well. However the other parts of the hand are very poorly tracked. For lower values of $\sigma$ the contour distribution did not track

the hand, but stayed in roughly the position of the initial contour distribution. For higher values of $\sigma$, the contour looped around in the general area of the fingers.

Figure 2 shows the contour distribution for frame 100 and 12 particles, for different numbers of iterations and values of $\sigma$. When $\sigma = 1$ and 2 the contour distribution does not keep up with the deformation. For $\sigma = 4$ the contour is correctly tracked except for the case of a single iteration. The rejection ratio (i.e. the ratio of rejected proposal particles to the total number of proposed particles) is shown as a bar on the right side of each image. Notice that the general trend is that rejection ratio increases as $\sigma$ increases, and decreases as the number of iterations is increased (due to a smaller $\sigma$ at each step).

Intuitively, it is not surprising that our new algorithm outperforms standard condensation. In the case of condensation, Gaussian noise is added to each control point at each time step. One particle may be correctly positioned for the little finger and poorly positioned for the forefinger, whereas an other particle may be well positioned around the forefinger and poorly positioned around the little finger. In order to track the deformation of the hand, some particles are required that track both the little finger and the forefinger (and all other parts too). In contrast the Metropolis updates are likely to reject particles that are locally worse than the current particle, but accept local improvements.

It should be noted that for lower dimensional problems, the increase in tracking performance is not as dramatic. E.g. in the case of tracking a rotating head, using a 12 control point b-spline, the two algorithms performed comparably.

## 4   Future work and conclusion

We are currently examining the effects of background clutter on the performance of the algorithm. We are also investigating other sequences and groupings of control points for generating proposal particles, and ways of using subspace models in combination with Metropolis updates.

In this paper we showed how Metropolis updates can be used to keep highly flexible active contours on track, and an efficient implementation strategy was presented. For high dimensional problems which are common for detailed shapes, the new algorithm presented produces dramatically better results than standard condensation.

### Acknowledgments

We thank Andrew Blake and Dale Schuurmans for helpful discussions.

## Footnotes

[1]Notation: We will use $Y$ to refer to a curve, parameterized by $x$, and $Y(s)$ for a particular point on the curve. $x$ refers to a particle consisting of subspace parameters, or in our case, control points. $n$ indexes a particle in a particle set, $i$ indexes a component of a particle (i.e. a single control point), $m$ indexes measurement lines and $t$ is used as a frame index

## References

[1] J. Y. A. Wang and E. H. Adelson "Representing moving images with layers." *IEEE Transactions on Image Processing, Special Issue: Image Sequence Compression*, vol. 3, no. 5. 1994, pp 625–638

[2] Y. Weiss "Smoothness in layers: Motion segmentation using nonparametric mixture estimation." *Proceedings of IEEE conference on Computer Vision and Pattern Recognition,* 1997.

[3] A. Jepson and M. J. Black "Mixture models for optical flow computation." *Proceedings of the IEEE Conference on Computer Vision and Pattern Recognition.*

[4] W. T. Freeman and P. A. Viola "Bayesian model of surface perception." *Advances in Neural Information Processing Systems* 10, MIT Press, 1998.

[5] W. Freeman, E. Pasztor,"Learning low-level vision," *Proceedings of the International Conference on Computer Vision*, 1999 pp. 1182–1189

[6] N. R. Howe, M. E. Leventon, W. T. Freeman, "Bayesian Reconstruction of 3D Human Motion from Single-Camera Video To appear in:" *Advances in Neural Information Processing Systems* 12, edited by S. A. Solla, T. K. Leen, and K-R Muller, 2000. TR99-37.

[7] G. E. Hinton, Z. Ghahramani and Y. W. Teh "Learning to parse images." In S.A. Solla, T. K. Leen, and K.-R. Müller (eds) *Advances in Neural Information Processing Systems 12*, MIT Press, 2000

[8] D. Terzopoulos, R. Szeliski, " Tracking with Kalman snakes" In A. Blake and A. Yuille (ed) *Active Vision,* 3–20. MIT Press, Cambridge, MA, 1992

[9] N. Papanikolopoulos, P. Khosla, T. Kanade "Vision and Control Techniques for robotic visual tracking," *In Proc. IEEE Int. Conf. Robotics and Autmation* 1, 1991, pp. 851 – 856.

[10] A. Blake, M. Isard "Active Contours" Springer-Verlag 1998 ISBN 3540762175

[11] J. MacCormick, A. Blake "A probabilistic exclusion principle for tracking multiple objects" *Proc. 7th IEEE Int. Conf. Computer Vision,* 1999

[12] M. Isard, A. Blake "ICONDENSATION: Unifying low-level and high-level tracking in a stochastic framework" *Proc. 5th European Conf. Computer Vision,* vol. 1 1998, pp. 893–908

[13] J. Sullivan, A. Blake, M. Isard, J. MacCormick, "Object Localization by Bayesian Correlation" *Proc. Int. Conf. Computer Vision,* 1999

[14] T. F. Cootes, G. H. Edwards, C. J. Taylor, "Active Appearance Models" *Proceedings of the European conference on Computer Vision*, Vol. 2, 1998, pp. 484 – 498

[15] I. Matthews, J. A. Bangham, R. Harvey and S. Cox. *Proc. Auditory-Visual Speech Processing (AVSP)*, 1998 pp. 73–78.

[16] R. M. Neal, "Probabilistic Inference Using Markov Chain Monte Carlo Methods", Technical Report CRG-TR-93-1, University of Toronto, 1993

[17] D. J. C MacKay "Introduction to Monte Carlo methods" In M. I. Jordan (ed) *Learning in Graphical Models*, MIT Press, Cambridge, MA, 1999
